# Emergence of Global Structure from Local Associations

**Thea B. Ghiselli-Crippa**
Department of Information Science
University of Pittsburgh
Pittsburgh PA 15260

**Paul W. Munro**
Department of Information Science
University of Pittsburgh
Pittsburgh PA 15260

## ABSTRACT

A variant of the encoder architecture, where units at the input and output layers represent nodes on a graph, is applied to the task of mapping locations to sets of neighboring locations. The degree to which the resulting internal (i.e. hidden unit) representations reflect global properties of the environment depends upon several parameters of the learning procedure. Architectural bottlenecks, noise, and incremental learning of landmarks are shown to be important factors in maintaining topographic relationships at a global scale.

## 1 INTRODUCTION

The acquisition of spatial knowledge by exploration of an environment has been the subject of several recent experimental studies, investigating such phenomena as the relationship between distance estimation and priming (e.g. McNamara et al., 1989) and the influence of route information (McNamara et al., 1984). Clayton and Habibi (1991) have gathered data suggesting that temporal contiguity during exploration is an important factor in determining associations between spatially distinct sites. This data supports the notion that spatial associations are built by a temporal process that is active during exploration and by extension supports Hebb's (1949) neurophysiological postulate that temporal associations underlie mechanisms of synaptic learning. Local spatial information acquired during the exploration process is continuously integrated into a global representation of the environment (cognitive map), which is typically arrived at by also considering global constraints, such as low dimensionality, not explicitly represented in the local relationships.

## 2  NETWORK ARCHITECTURE AND TRAINING

The goal of this network design is to reveal structure among the internal representations that emerges solely from integration of local spatial associations; in other words, to show how a network trained to learn only local spatial associations characteristic of an environment can develop internal representations which capture global spatial properties. A variant of the encoder architecture (Ackley et al., 1985) is used to associate each node on a 2-D graph with the set of its neighboring nodes, as defined by the arcs in the graph. This 2-D neighborhood mapping task is similar to the 1-D task explored by Wiles (1993) using an N-2-N architecture, which can be characterized in terms of a graph environment as a circular chain with broad neighborhoods.

In the neighborhood mapping experiments described in the following, the graph nodes are visited at random: at each iteration, a training pair (node-neighborhood) is selected at random from the training set. As in the standard encoder task, the input patterns are all or-thogonal, so that there is no structure in the input domain that the network could exploit in constructing the internal representations; the only information about the structure of the environment comes from the local associations that the network is shown during training.

### 2.1  N-H-N NETWORKS

The neighborhood mapping task was first studied using a strictly layered feed-forward N-H-N architecture, where N is the number of input and output units, corresponding to the number of nodes in the environment, and H is the number of units in the single hidden layer. Experiments were done using square grid environments with wrap-around (toroidal) and without wrap-around (bounded) at the edges. The resulting hidden unit representations reflect global properties of the environment to the extent that distances between them correlate with distances between corresponding points on the grid. These two distance measures are plotted against one another in Figure 1 for toroidal and bounded environments.

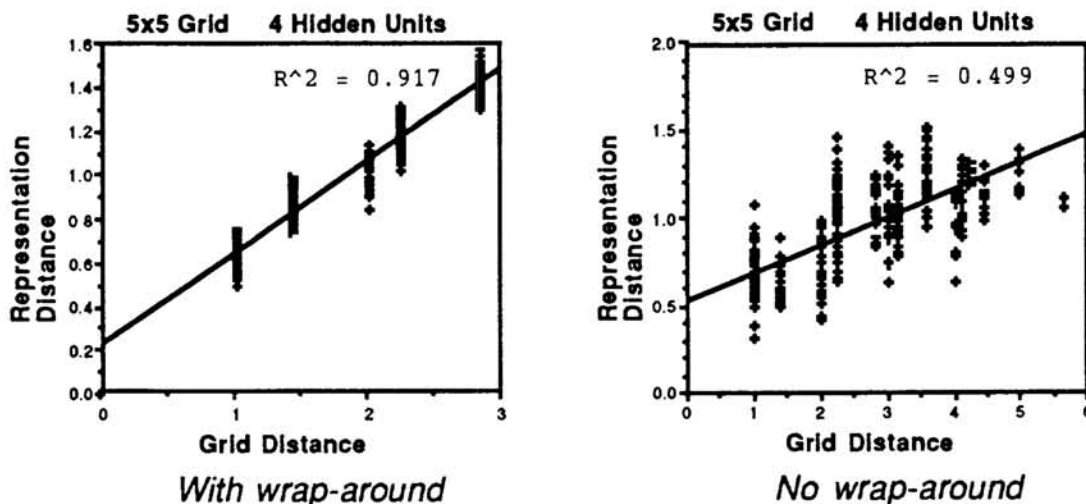

Figure 1: Scatterplots of Distances between Hidden Unit Representations vs. Distances between Corresponding Locations in the Grid Environment.

## 2.2  N-2-H-N Networks

A hidden layer with just two units forces representations into a 2-D space, which matches the dimensionality of the environment. Under this constraint, the image of the environment in the 2-D space may reflect the topological structure of the environment. This conjecture leads to a further conjecture that the 2-D representations will also reveal global relationships of the environment. Since the neighborhoods in a 2-D representation are not linearly separable regions, another layer (H-layer) is introduced between the two-unit layer and the output (see Figure 2). Thus, the network has a strictly layered feed-forward N-2-H-N architecture, where the N units at the input and output layers correspond to the N nodes in the environment, two units make up the topographic layer, and H is the number of units chosen for the new layer (H is estimated according to the complexity of the graph). Responses for the hidden units (in both the T- and H-layers) are computed using the hyperbolic tangent (which ranges from -1 to +1), while the standard sigmoid (0 to +1) is used for the output units, to promote orthogonality between representations (Munro, 1989). Instead of the squared error, the cross entropy function (Hinton, 1987) is used to avoid problems with low derivatives observed in early versions of the network.

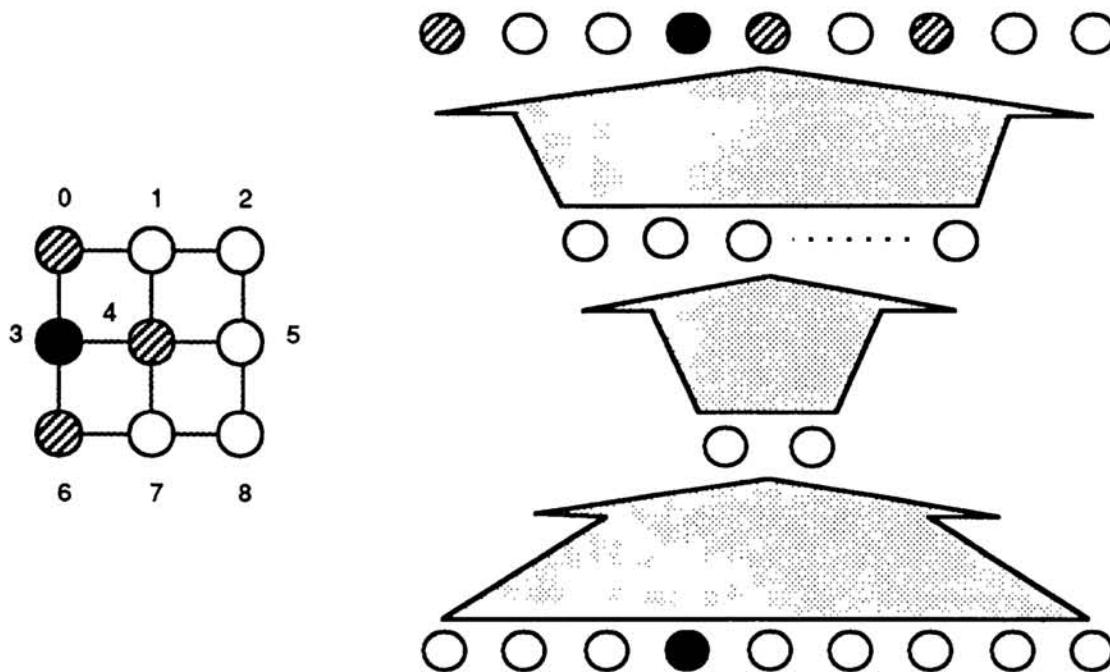

Figure 2: A 3x3 Environment and the Corresponding Network. When input unit 3 is activated, the network responds by activating the same unit and all its neighbors.

## 3  RESULTS

### 3.1  T-UNIT  RESPONSES

Neighborhood mapping experiments were done using bounded square grid environments and N-2-H-N networks. After training, the topographic unit activities corresponding to each of the N possible inputs are plotted, with connecting lines representing the arcs from

the environment. Each axis in Figure 3 represents the activity of one of the T-units. These maps can be readily examined to study the relationship between their global structure and the structure of the environment. The receptive fields of the T-units give an alternative representation of the same data: the response of each T-unit to all N inputs is represented by N circles arranged in the same configuration as the nodes in the grid environment. Circle size is proportional to the absolute value of the unit activity; filled circles indicate negative values, open circles indicate positive values. The receptive field represents the T-unit's sensitivity with respect to the environment.

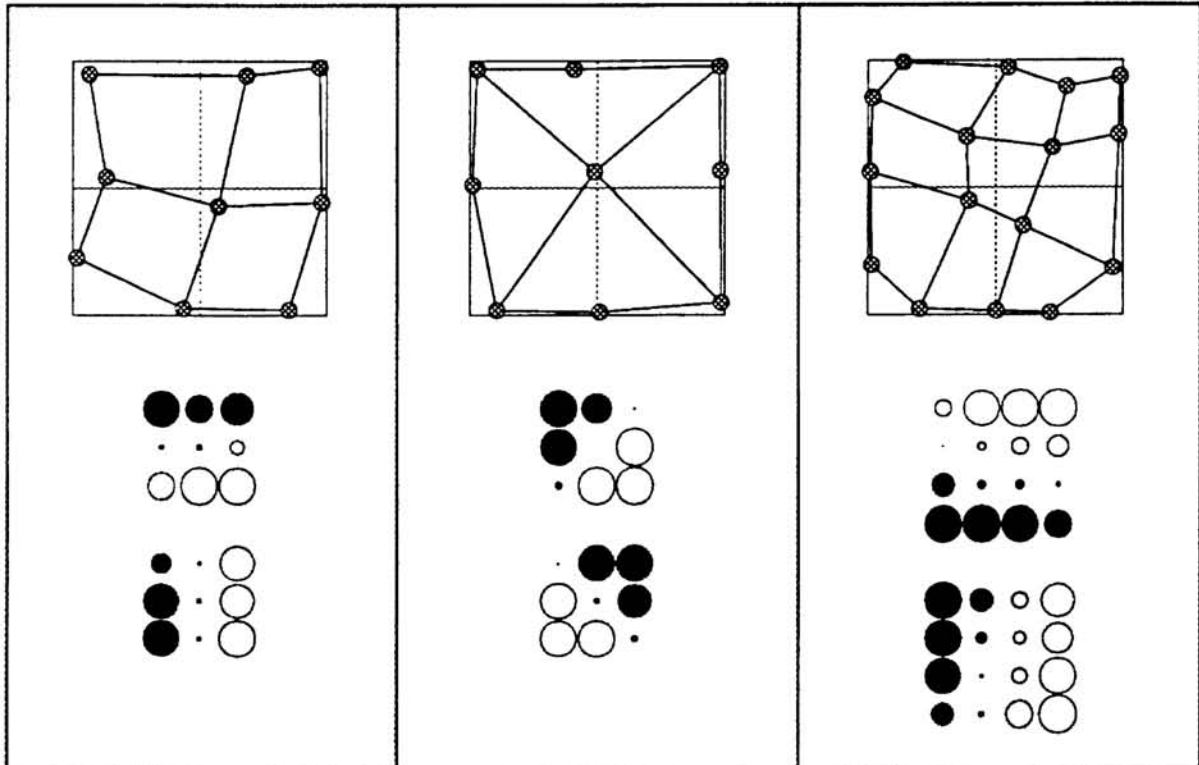

Figure 3: Representations at the Topographic Layer. Activity plots and receptive fields for two 3x3 grids (left and middle) and a 4x4 grid(right).

The two 3x3 cases shown in Figure 3 illustrate alternative solutions that are each locally consistent, but have different global structure. In the first case, it is evident how the first unit is sensitive to changes in the vertical location of the grid nodes, while the second unit is sensitive to their horizontal location. The axes are essentially rotated 45 degrees in the second case. Except for this rotation of the reference axes, both representations captured the global structure of the 3x3 environment.

## 3.2   NOISE IN THE HIDDEN UNITS

While networks tended to form maps in the T-layer that reflect the global structure of the environment, in some cases the maps showed correspondences that were less obvious: i.e., the grid lines crossed, even though the network converged. A few techniques have proven valuable for promoting global correspondence between the topographic representations and the environment, including Judd and Munro's (1993) introduction of noise as pressure to separate representations. The noise is implemented as a small probability for

reversing the sign of individual H-unit outputs. As reported in a previous study (Ghiselli-Crippa and Munro, 1994), the presence of noise causes the network to develop topographic representations which are more separated, and therefore more robust, so that the correct output units can be activated even if one or more of the H-units provides an incorrect output. From another point of view, the noise can be seen as causing the network to behave as if it had an effective number of hidden units which is smaller than the given number H. The introduction of noise as a means to promote robust topographic representations can be appreciated by examining Figure 4, which illustrates the representations of a 5x5 grid developed by a 25-2-20-25 network trained without noise (left) and with noise (middle) (the network was initialized with the same set of small random weights in all cases). Note that the representations developed by the network subject to noise are more separated and exhibit the same global structure as the environment. To avoid convergence problems observed with the use of noise throughout the whole training process, the noise can be introduced at the beginning of training and then gradually reduced over time.

A similar technique involves the use of low-level noise injected in the T-layer to directly promote the formation of well-separated representations. Either Gaussian or uniform noise directly added to the T-unit outputs gives comparable results. The use of noise in either hidden layer has a beneficial influence on the formation of globally consistent representations. However, since the noise in the H-units exerts only an indirect influence on the T-unit representations, the choice of its actual value seems to be less crucial than in the case where the noise is directly applied at the T-layer.

The drawback for the use of noise is an increase in the number of iterations required by the network to converge, that scales up with the magnitude and duration of the noise.

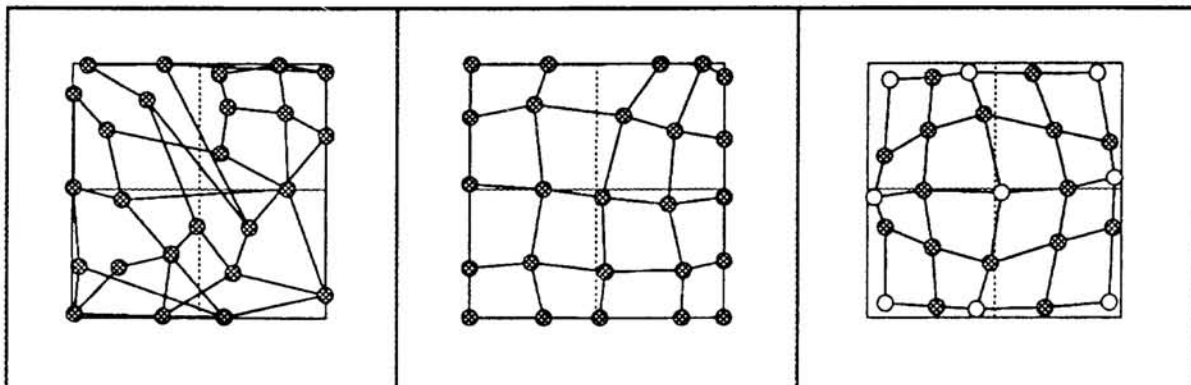

Figure 4: Representations at the Topographic Layer. Training with no noise (left) and with noise in the hidden units (middle); training using landmarks (right).

## 3.3   LANDMARK LEARNING

Another effective method involves the organization of training in 2 separate phases, to model the acquisition of landmark information followed by the development of route and/or survey knowledge (Hart and Moore, 1973; Siegel and White, 1975). This method is implemented by manipulating the training set during learning, using coarse spatial resolution at the outset and introducing interstitial features as learning progresses to the second phase. The first phase involves training the network only on a subset of the possible

N patterns (landmarks). Once the landmarks have been learned, the remaining patterns are added to the training set. In the second phase, training proceeds as usual with the full set of training patterns; the only restriction is applied to the landmark points, whose topographical representations are not allowed to change (the corresponding weights between input units and T-units are frozen), thus modeling the use of landmarks as stable reference points when learning the details of a new environment. The right pane of Figure 4 illustrates the representations developed for a 5x5 grid using landmark training; the same 25-2-20-25 network mentioned above was trained in 2 phases, first on a subset of 9 patterns (landmarks) and then on the full set of 25 patterns (the landmarks are indicated as white circles in the activity plot).

## 3.4   NOISE IN LANDMARK LEARNING

The techniques described above (noise and landmark learning) can be combined together to better promote the emergence of well-structured representation spaces. In particular, noise can be used during the first phase of landmark learning to encourage a robust representation of the landmarks: Figure 5 illustrates the representations obtained for a 5x5 grid using landmark training with two different levels of noise in the H-units during the first phase. The effect of noise is evident when comparing the 4 corner landmarks in the right pane of Figure 4 (landmark learning with no noise) with those in Figure 5. With increasing levels of noise, the T-unit activities corresponding to the 4 corner landmarks approach the asymptotic values of +1 and -1; the activity plots illustrate this effect by showing how the corner landmark representations move toward the corners of T-space, reaching a configuration which provides more resistance to noise. During the second phase of training, the landmarks function as reference points for the additional features of the environment and their positioning in the representational space therefore becomes very important. A well-formed, robust representation of the landmarks at the end of the first phase is crucial for the formation of a map in T-space that reflects global structure, and the use of noise can help promote this.

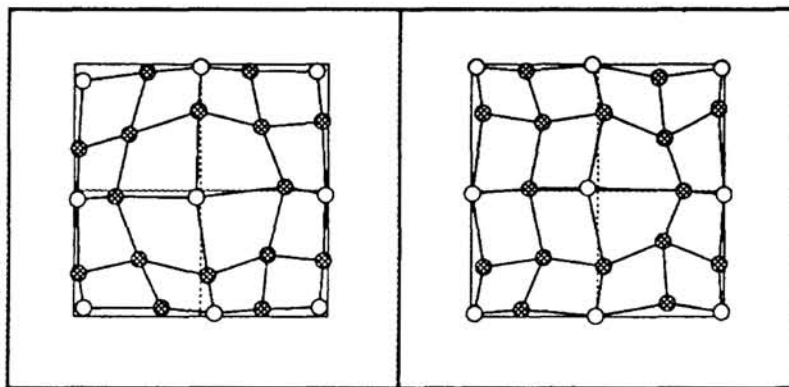

Figure 5:  Representations at the Topographic Layer.  Landmark training using noise in phase 1: low noise level (left), high noise level (right).

## 4   DISCUSSION

Large scale constraints intrinsic to natural environments, such as low dimensionality, are not necessarily reflected in local neighborhood relations, but they constitute information which is essential to the successful development of useful representations of the environ-

ment. In our model, some of the constraints imposed on the network architecture effec-tively reduce the dimensionality of the representational space. Constraints have been in-troduced several ways: bottlenecks, noise, and landmark learning; in all cases, these con-straints have had constructive influences on the emergence of globally consistent repre-sentation spaces. The approach described presents an alternative to Kohonen's (1982) scheme for capturing topography; here, topographic relations emerge in the representa-tional space, rather than in the weights between directly connected units.

The experiments described thus far have focused on how global spatial structure can emerge from the integration of local associations and how it is affected by the introduc-tion of global constraints. As mentioned in the introduction, one additional factor influ-encing the process of acquisition of spatial knowledge needs to be considered: temporal contiguity during exploration, that is, how temporal associations of spatially adjacent lo-cations can influence the representation of the environment. For example, a random type of exploration ("wandering") can be considered, where the next node to be visited is select-ed at random from the neighbors of the current node. Preliminary studies indicate that such temporal contiguity during training results in the formation of hidden unit represen-tations with global properties qualitatively similar to those reported here. Alternatively, more directed exploration methods can be studied, with a systematic pattern guiding the choice of the next node to be visited. The main purpose of these studies will be to show how different exploration strategies can affect the formation and the characteristics of cog-nitive maps of the environment.

Higher order effects of temporal and spatial contiguity can also be considered. However, in order to capture regularities in the training process that span several exploration steps, simple feed-forward networks may no longer be sufficient; partially recurrent networks (Elman, 1990) are a likely candidate for the study of such processes.

## Acknowledgements

We wish to thank Stephen Hirtle, whose expertise in the area of spatial cognition greatly benefited our research. We are also grateful for the insightful comments of Janet Wiles.

## References

D. H. Ackley, G. E. Hinton, and T. J. Sejnowski (1985) "A learning algorithm for Boltzmann machines," *Cognitive Science*, vol. 9, pp. 147-169.

K. Clayton and A. Habibi (1991) "The contribution of temporal contiguity to the spatial priming effect," *Journal of Experimental Psychology: Learning, Memory, and Cognition*, vol. 17, pp. 263-271.

J. L. Elman (1990) "Finding structure in time," *Cognitive Science*, vol. 14, pp. 179-211.

T. B. Ghiselli-Crippa and P. W. Munro (1994) "Learning global spatial structures from local associations," in M. C Mozer, P. Smolensky, D. S. Touretzky, J. L. Elman, and A. S. Weigend (Eds.), *Proceedings of the 1993 Connectionist Models Summer School*, Hillsdale, NJ: Erlbaum.

R. A. Hart and G. T. Moore (1973) "The development of spatial cognition: A review," in R. M. Downs and Stea (Eds.), *Image and Environment*, Chicago, IL: Aldine.

D. O. Hebb (1949) *The Organization of Behavior*, New York, NY: Wiley.

G. E. Hinton (1987) "Connectionist learning procedures," *Technical Report CMU-CS-87-115, version 2*, Pittsburgh, PA: Carnegie-Mellon University, Computer Science Department.

S. Judd and P. W. Munro (1993) "Nets with unreliable hidden nodes learn error-correcting codes," in C. L. Giles, S. J. Hanson, and J. D. Cowan, *Advances in Neural Information Processing Systems 5*, San Mateo, CA: Morgan Kaufmann.

T. Kohonen (1982) "Self-organized formation of topological correct feature maps," *Biological Cybernetics*, vol. 43, pp. 59-69.

T. P. McNamara, J. K. Hardy, and S. C. Hirtle (1989) "Subjective hierarchies in spatial memory," *Journal of Experimental Psychology: Learning, Memory, and Cognition*, vol. 15, pp. 211-227.

T. P. McNamara, R. Ratcliff, and G. McKoon (1984) "The mental representation of knowledge acquired from maps," *Journal of Experimental Psychology: Learning, Memory, and Cognition*, vol. 10, pp. 723-732.

P. W. Munro (1989) "Conjectures on representations in backpropagation networks," *Technical Report TR-89-035*, Berkeley, CA: International Computer Science Institute.

A. W. Siegel and S. H. White (1975) "The development of spatial representations of large-scale environments," in H. W. Reese (Ed.), *Advances in Child Development and Behavior*, New York, NY: Academic Press.

J. Wiles (1993) "Representation of variables and their values in neural networks," in *Proceedings of the Fifteenth Annual Conference of the Cognitive Science Society*, Hillsdale, NJ: Erlbaum.